# Prediction of Beta Sheets in Proteins

**Anders Krogh**
The Sanger Centre
Hinxton, Cambs CB10 1RQ, UK.
*Email: krogh@sanger.ac.uk*

**Søren Kamaric Riis**
Electronics Institute, Building 349
Technical University of Denmark
2800 Lyngby, Denmark
*Email: riis@ei.dtu.dk*

## Abstract

Most current methods for prediction of protein secondary structure use a small window of the protein sequence to predict the structure of the central amino acid. We describe a new method for prediction of the non-local structure called $\beta$-sheet, which consists of two or more $\beta$-strands that are connected by hydrogen bonds. Since $\beta$-strands are often widely separated in the protein chain, a network with two windows is introduced. After training on a set of proteins the network predicts the sheets well, but there are many false positives. By using a global energy function the $\beta$-sheet prediction is combined with a local prediction of the three secondary structures $\alpha$-helix, $\beta$-strand and coil. The energy function is minimized using simulated annealing to give a final prediction.

## 1 INTRODUCTION

Proteins are long sequences of amino acids. There are 20 different amino acids with varying chemical properties, *e.g.*, some are hydrophobic (dislikes water) and some are hydrophilic [1]. It is convenient to represent each amino acid by a letter and the sequence of amino acids in a protein (the *primary structure*) can be written as a string with a typical length of 100 to 500 letters. A protein chain folds back on itself, and the resulting 3D structure (the *tertiary structure*) is highly correlated to the function of the protein. The prediction of the 3D structure from the primary structure is one of the long-standing unsolved problems in molecular biology. As an important step on the way a lot of work has been devoted to predicting the local conformation of the protein chain, which is called the *secondary structure*. Neural network methods are currently the most successful for predicting secondary structure. The approach was pioneered by Qian and Sejnowski [2] and Bohr *et al.* [3], but later extended in various ways, see *e.g.* [4] for an overview. In most of this work, only the two regular secondary structure elements $\alpha$-helix and $\beta$-strand are being distinguished, and everything else is labeled coil. Thus, the methods based

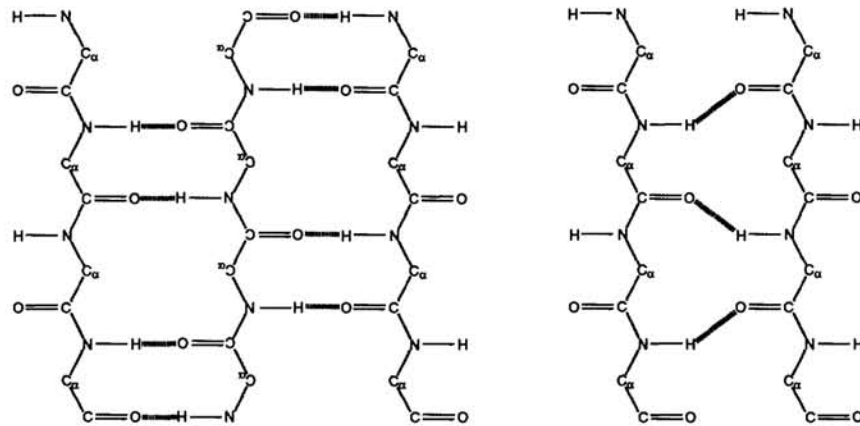

Figure 1: Left: Anti-parallel $\beta$-sheet. The vertical lines correspond to the backbone of the protein. An amino acid consists of N–C$_\alpha$–C and a side chain on the C$_\alpha$ that is not shown (the 20 amino acids are distinguished by different side chains). In the anti-parallel sheet the directions of the strands alternate, which is here indicated quite explicitly by showing the middle strand up-side down. The H-bonds between the strands are shown by ‖‖‖‖. A sheet has two or more strands, here the anti-parallel sheet is shown with three strands. Right: Parallel $\beta$-sheet consisting of two strands.

on a local window of amino acids give a three-state prediction of the secondary structure of the central amino acid in the window.

Current predictions of secondary structure based on single sequences as input have accuracies of about 65-66%. It is widely believed that this accuracy is close to the limit of what can be done from a local window (using only single sequences as input) [5], because interactions between amino acids far apart in the protein chain are important to the structure. A good example of such non-local interactions are the $\beta$-sheets consisting of two or more $\beta$-strands interconnected by H-bonds, see fig. 1. Often the $\beta$-strands in a sheet are widely separated in the sequence, implying that only part of the available sequence information about a $\beta$-sheet can be contained in a window of, say, 13 amino acids. This is one of the reasons why the accuracy of $\beta$-strand predictions are generally lower than the accuracy of $\alpha$-helix predictions. The aim of this work is to improve prediction of secondary structures by combining local predictions of $\alpha$-helix, $\beta$-strand and coil with a non-local method predicting $\beta$-sheets.

Other work along the same directions include [6] in which $\beta$-sheet predictions are done by linear methods and [7] where a so-called density network is applied to the problem.

## 2   A NEURAL NETWORK WITH TWO WINDOWS

We aim at capturing correlations in the $\beta$-sheets by using a neural network with two windows, see fig. 2. While window 1 is centered around amino acid number $i$ ($a_i$), window 2 slides along the rest of the chain. When the amino acids centered in each of the two windows sit opposite each other in a $\beta$-sheet the target output is 1, and otherwise 0. After the whole protein has been traversed by window 2, window 1 is moved to the next position ($i+1$) and the procedure is repeated. If the protein is $L$ amino acids long this procedure yields an output value for each of the $L(L-1)/2$

Figure 2: Neural network for predicting $\beta$-sheets. The network employs weight sharing to improve the encoding of the amino acids and to reduce the number of adjustable parameters.

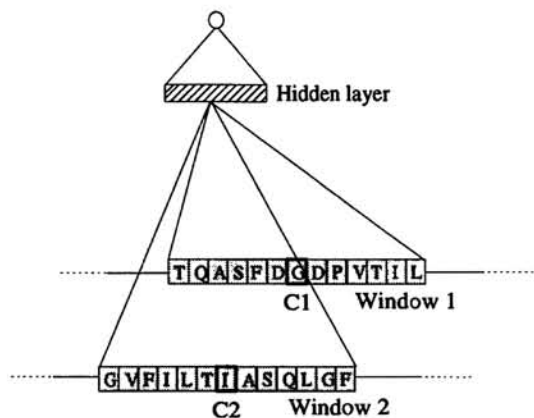

pairs of amino acids. We display the output in a $L \times L$ gray-scale image as shown in fig. 3. We assume symmetry of sheets, *i.e.*, if the two windows are interchanged, the output does not change. This symmetry is ensured (approximately) during training by presenting all inputs in both directions.

Each window of the network sees $K$ amino acids. An amino acid is represented by a vector of 20 binary numbers all being zero, except one, which is 1. That is, the amino acid A is represented by the vector $1, 0, 0, \ldots, 0$ and so on. This coding ensures that the input representations are uncorrelated, but it is a very inefficient coding, since 20 amino acids could in principle be represented by only 5 bit. Therefore, we use *weight sharing* [8] to learn a better encoding [4]. The 20 input units corresponding to one window position are fully connected to three hidden units. The $3 \times (20 + 1)$ weights to these units are shared by all window positions, *i.e.*, the activation of the 3 hidden units is a new *learned* encoding of the amino acids, so instead of being represented by 20 binary values they are represented by 3 real values. Of course the number of units for this encoding can be varied, but initial experiments showed that 3 was optimal [4]. The two windows of the network are made the same way with the same number of inputs *etc.*. The first layer of hidden units in the two windows are fully connected to a hidden layer which is fully connected to the output unit, see fig. 2. Furthermore, two structurally identical networks are used: one for parallel and one for anti-parallel $\beta$-sheets.

The basis for the training set in this study is the set of 126 non-homologous protein chains used in [9], but chains forming $\beta$-sheets with *other* chains are excluded. This leaves us with 85 proteins in our data set. For a protein of length $L$ only a very small fraction of the $L(L-1)/2$ pairs are positive examples of $\beta$-sheet pairs. Therefore it is very important to balance the positive and negative examples to avoid the situation where the network always predicts no $\beta$-sheet. Furthermore, there are several types of negative examples with quite different occurrences: 1) two amino acids of which none belong to a $\beta$-sheet; 2) one in a $\beta$-sheet and one which is not in a $\beta$-sheet; 3) two sitting in $\beta$-sheets, but not opposite to each other. The balancing was done in the following way. For each positive example selected at random a negative example from each of the three categories were selected at random.

If the network does not have a second layer of hidden units, it turns out that the result is no better than a network with only one input window, *i.e.*, the network cannot capture correlations between the two windows. Initial experiments indicated that about 10 units in the second hidden layer and two identical input windows of size $K = 9$ gave the best results. In fig. 3(left) the prediction of anti-parallel sheets is shown for the protein identified as 1acx in the Brookhaven Protein Data Bank

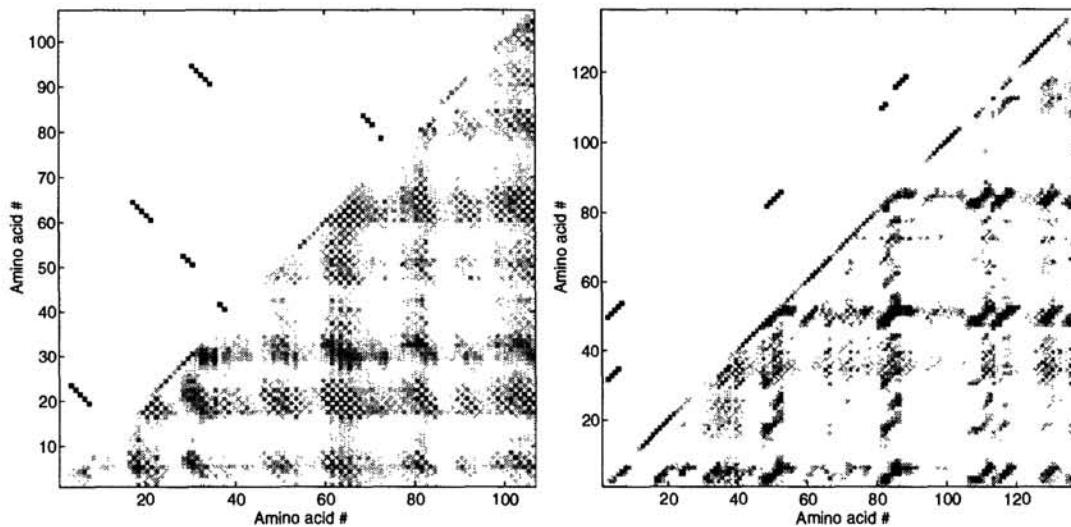

Figure 3: Left: The prediction of anti-parallel $\beta$-sheets in the protein 1acx. In the upper triangle the correct structure is shown by a black square for each $\beta$-sheet pair. The lower triangle shows the prediction by the two-window network. For any pair of amino acids the network output is a number between zero (white) and one (black), and it is displayed by a linear gray-scale. The diagonal shows the prediction of $\alpha$-helices. Right: The same display for parallel $\beta$-sheets in the protein 4fxn. Notice that the correct structure are lines parallel to the diagonal, whereas they are perpendicular for anti-parallel sheets. For both cases the network was trained on a training set that did *not* contain the protein for which the result is shown.

[10]. First of all, one notices the checker board structure of the prediction of $\beta$-sheets. This is related to the structure of $\beta$-sheets. Many sheets are hydrophobic on one side and hydrophilic on the other. The side chains of the amino acids in a strand alternates between the two sides of the sheet, and this gives rise to the periodicity responsible for the pattern.

Another network was trained on *parallel* $\beta$-sheets. These are rare compared to the anti-parallel ones, so the amount of training data is limited. In fig. 3(right) the result is shown for protein 4fxn. This prediction seems better than the one obtained for anti-parallel sheets, although false positive predictions still occurs at some positions with strands that do not pair. Strands that bind in parallel $\beta$-sheets are generally more widely separated in the sequence than strands in anti-parallel sheets. Therefore, one can imagine that the strands in parallel sheets have to be more correlated to find each other in the folding process, which would explain the better prediction accuracy.

The results shown in fig. 3 are fairly representative. The network misses some of the sheets, but false positives present a more severe problem. By calculating correlation coefficients we can show that the network *does* capture some correlations, but they seem to be weak. Based on these results, we hypothesize that the formation of $\beta$-sheets is only weakly dependent on correlations between corresponding $\beta$-strands. This is quite surprising. However weak these correlations are, we believe they can still improve the accuracy of the three state secondary structure prediction. In order to combine local methods with the non-local $\beta$-sheet prediction, we introduce a global energy function as described below.

## 3   A GLOBAL ENERGY FUNCTION

We use a newly developed local neural network method based on *one* input window [4] to give an initial prediction of the three possible structures. The output from this network is constrained by softmax [11], and can thus be interpreted as the probabilities for each of the three structures. That is, for amino acid $a_i$, it yields three numbers $p_{i,n}$, $n = 1, 2$ or $3$ indicating the probability of $\alpha$-helix ($p_{i,1}$), $\beta$-sheet ($p_{i,2}$), or coil ($p_{i,3}$). Define $s_{i,n} = 1$ if amino acid $i$ is assigned structure $n$ and $s_{i,n} = 0$ otherwise. Also define $h_{i,n} = \log p_{i,n}$. We now construct the 'energy function'

$$H_3(s) = -\sum_i \sum_n u_n h_{i,n} s_{i,n}, \tag{1}$$

where weights $u_n$ are introduced for later usage. Assuming the probabilities $p_{i,n}$ are independent for any two amino acids in a sequence, this is the negative log likelihood of the assigned secondary structure represented by $s$, provided that $u_n = 1$. As it stands, alone, it is a fairly trivial energy function, because the minimum is the assignment which corresponds to the prediction with the maximum $p_{i,n}$ at each position $i$ — the assignment of secondary structure that one would probably use anyway.

For amino acids $a_i$ and $a_j$ the logarithm of the output of the $\beta$-sheet network described previously is called $q_{ij}^p$ for parallel $\beta$-sheets and $q_{ij}^a$ for anti-parallel sheets. We interpret these numbers as the gain in energy if a $\beta$-sheet pair is formed. (As more terms are added to the energy, the interpretation as a log-likelihood function is gradually fading.) If the two amino acids form a pair in a parallel $\beta$-sheet, we set the variable $T_{ij}^p$ equal to 1, and otherwise to 0, and similarly with $T_{ij}^a$ for anti-parallel sheets. Thus the $T_{ij}^a$ and $T_{ij}^p$ are sparse binary matrices. Now the total energy of the $\beta$-sheets can be expressed as

$$H_\beta(s, T^a, T^p) = -\sum_{ij} [C_a q_{ij}^a T_{ij}^a + C_p q_{ij}^p T_{ij}^p], \tag{2}$$

where $C_a$ and $C_p$ determine the weights of the two terms in the function. Since an amino acid can only be in one structure, the dynamic $T$ and $s$ variables are constrained: Only $T_{ij}^a$ or $T_{ij}^p$ can be 1 for the same $(i, j)$, and if any of them is 1 the amino acids involved must be in a $\beta$-sheet, so $s_{i,2} = s_{j,2} = 1$. Also, $s_{i,2}$ can only be 1 if there exists a $j$ with either $T_{ij}^a$ or $T_{ij}^p$ equal to 1. Because of these constraints we have indicated an $s$ dependence of $H_\beta$.

The last term in our energy function introduces correlations between neighboring amino acids. The above assumption that the secondary structure of the amino acids are independent is of course a bad assumption, and we try to repair it with a term

$$H_n(s) = \sum_i \sum_{nm} J_{nm} s_{i,n} s_{i+1,m}, \tag{3}$$

that introduces nearest neighbor interactions in the chain. A negative $J_{11}$, for instance, means that $\alpha$ following $\alpha$ is favored, and *e.g.*, a positive $J_{12}$ discourages a $\beta$ following an $\alpha$.

Now the total energy is

$$H_{\text{total}}(s, T^a, T^p) = H_3(s) + H_\beta(s, T^a, T^p) + H_n(s). \tag{4}$$

Since $\beta$-sheets are introduced in two ways, through $h_{i,2}$ and $q_{ij}$, we need the weights $u_n$ in (1) to be different from 1.

The total energy function (4) has some resemblance with a so-called Potts glass in an external field [12]. The crucial difference is that the couplings between the

'spins' $s_i$ are dependent on the dynamic variables $T$. Another analogy of the energy function is to image analysis, where couplings like the $T$'s are sometimes used as edge elements.

## 3.1  PARAMETER ESTIMATION

The energy function contains a number of parameters, $u_n$, $C_a$, $C_p$ and $J_{nm}$. These parameters were estimated by a method inspired by Boltzmann learning [13]. In the Boltzmann machine the estimation of the weights can be formulated as a minimization of the difference between the free energy of the 'clamped' system and that of the 'free-running' system [14]. If we think of our energy function as a free energy (at zero temperature), it corresponds to minimizing the difference between the energy of the correct protein structure and the minimum energy,

$$C = \sum_{\mu=1}^{p} \left[ H_{\text{total}}\left(s(\mu), T^a(\mu), T^p(\mu)\right) - H_{\text{total}}\left(\hat{s}(\mu), \hat{T}^a(\mu), \hat{T}^p(\mu)\right)\right], \qquad (5)$$

where $p$ is the total number of proteins in the training set. Here the *correct* structure of protein $\mu$ is called $s(\mu), T^a(\mu), T^p(\mu)$, whereas $\hat{s}(\mu), \hat{T}^a(\mu), \hat{T}^p(\mu)$ represents the structure that minimizes the energy $H_{\text{total}}$. By definition the second term of $C$ is less than the first, so $C$ is bounded from below by zero.

The cost function $C$ is minimized by gradient descent in the parameters. This is in principle straightforward, because all the parameters appear linearly in $H_{\text{total}}$. However, a problem with this approach is that $C$ is minimal when all the parameters are set to zero, because then the energy is zero. It is cured by constraining some of the parameters in $H_{\text{total}}$. We chose the constraint $\sum_n u_n = 1$. This may not be the perfect solution from a theoretical point of view, but it works well. Another problem with this approach is that one has to find the minimum of the energy $H_{\text{total}}$ in the dynamic variables in each iteration of the gradient descent procedure. To globally minimize the function by simulated annealing each time would be very costly in terms of computer time. Instead of using the (global) minimum of the energy for each protein, we use the energy obtained by minimizing the energy from the correct structure. This minimization is done by a greedy algorithm in the following way. In each iteration the change in $s, T^a, T^p$ which results in the largest decrease in $H_{\text{total}}$ is carried out. This is repeated until any change will increase $H_{\text{total}}$. This algorithm works towards a *local stability* of the protein structures in the training set. We believe it is not only an efficient way of doing it, but also a very sensible way. In fact, the method may well be applicable in other models, such as Boltzmann machines.

## 3.2  STRUCTURE PREDICTION BY SIMULATED ANNEALING

After estimation of the parameters on which the energy function $H_{\text{total}}$ depends, we can proceed to predict the structure of new proteins. This was done using simulated annealing and the EBSA package [15]. The total procedure for prediction is,

1. A neural net predicts $\alpha$-helix, $\beta$-strand or coil. The logarithm of these predictions give all the $h_{i,n}$ for that protein.
2. The two-window neural networks predict the $\beta$-sheets. The result is the $q_{ij}^a$ from one network and the $q_{ij}^p$ from the other.
3. A random configuration of $s, T^a, T^p$ variables is generated from which the simulated annealing minimization of $H_{\text{total}}$ was started. During annealing, all constraints on $s, T^a, T^p$ variables are strictly enforced.

4. The final minimum configuration $\hat{s}$ is the prediction of the secondary structure. The $\beta$-sheets are predicted by $\hat{T}^a$ and $\hat{T}^p$.

Using the above scheme, an average secondary structure accuracy of 66.5% is obtained by seven-fold cross validation. This should be compared to 66.3% obtained by the local neural network based method [4] on the same data set. Although these preliminary results do not represent a significant improvement, we consider them very encouraging for future work. Because the method not only predicts the secondary structure, but also which strands actually binds to form $\beta$-sheets, even a modest result may be an important step on the way to full 3D predictions.

## 4 CONCLUSION

In this paper we introduced several novel ideas which may be applicable in other contexts than prediction of protein structure. Firstly, we described a neural network with two input windows that was used for predicting the non-local structure called $\beta$-sheets. Secondly, we combined local predictions of $\alpha$-helix, $\beta$-strand and coil with the $\beta$-sheet prediction by minimization of a global energy function. Thirdly, we showed how the adjustable parameters in the energy function could be estimated by a method similar to Boltzmann learning.

We found that correlations between $\beta$-strands in $\beta$-sheets are surprisingly weak. Using the energy function to combine predictions improves performance a little. Although we have not solved the protein folding problem, we consider the results very encouraging for future work. This will include attempts to improve the performance of the two-window network as well as experimenting with the energy function, and maybe add more terms to incorporate new constraints.

**Acknowledgments**: We would like to thank Tim Hubbard, Richard Durbin and Benny Lautrup for interesting comments on this work and Peter Salamon and Richard Frost for assisting with simulated annealing. This work was supported by a grant from the Novo Nordisk Foundation.

## References

[1] C. Branden and J. Tooze, *Introduction to Protein Structure* (Garland Publishing, Inc., New York, 1991).

[2] N. Qian and T. Sejnowski, Journal of Molecular Biology **202**, 865 (1988).

[3] H. Bohr *et al.*, FEBS Letters **241**, 223 (1988).

[4] S. Riis and A. Krogh, Nordita Preprint 95/34 S, submitted to J. Comp. Biol.

[5] B. Rost, C. Sander, and R. Schneider, J Mol. Biol. **235**, 13 (1994).

[6] T. Hubbard, in *Proc. of the 27th HICSS*, edited by R. Lathrop (IEEE Computer Soc. Press, 1994), pp. 336–354.

[7] D. J. C. MacKay, in *Maximum Entropy and Bayesian Methods, Cambridge 1994*, edited by J. Skilling and S. Sibisi (Kluwer, Dordrecht, 1995).

[8] Y. Le Cun *et al.*, Neural Computation 1, 541 (1989).

[9] B. Rost and C. Sander, Proteins **19**, 55 (1994).

[10] F. Bernstein *et al.*, J Mol. Biol. **112**, 535 (1977).

[11] J. Bridle, in *Neural Information Processing Systems 2*, edited by D. Touretzky (Morgan Kaufmann, San Mateo, CA, 1990), pp. 211–217.

[12] K. Fisher and J. Hertz, *Spin glasses* (Cambridge University Press, 1991).

[13] D. Ackley, G. Hinton, and T. Sejnowski, Cognitive Science **9**, 147 (1985).

[14] J. Hertz, A. Krogh, and R. Palmer, *Introduction to the Theory of Neural Computation* (Addison-Wesley, Redwood City, 1991).

[15] R. Frost, SDSC EBSA, C Library Documentation, version 2.1. SDSC Techreport.
